# Information Maximization in Single Neurons

**Martin Stemmler and Christof Koch**
Computation and Neural Systems Program
Caltech 139-74
Pasadena, CA 91125
Email: stemmler@klab.caltech.edu, koch@klab.caltech.edu

## Abstract

Information from the senses must be compressed into the limited range of firing rates generated by spiking nerve cells. Optimal compression uses all firing rates equally often, implying that the nerve cell's response matches the statistics of naturally occurring stimuli. Since changing the voltage-dependent ionic conductances in the cell membrane alters the flow of information, an unsupervised, non-Hebbian, developmental learning rule is derived to adapt the conductances in Hodgkin-Huxley model neurons. By maximizing the rate of information transmission, each firing rate within the model neuron's limited dynamic range is used equally often.

An efficient neuronal representation of incoming sensory information should take advantage of the regularity and scale invariance of stimulus features in the natural world. In the case of vision, this regularity is reflected in the typical probabilities of encountering particular visual contrasts, spatial orientations, or colors [1]. Given these probabilities, an optimized neural code would eliminate any redundancy, while devoting increased representation to commonly encountered features.

At the level of a single spiking neuron, information about a potentially large range of stimuli is compressed into a finite range of firing rates, since the maximum firing rate of a neuron is limited. Optimizing the information transmission through a single neuron in the presence of uniform, additive noise has an intuitive interpretation: the most efficient representation of the input uses every firing rate with *equal* probability. An analogous principle for non-spiking neurons has been tested experimentally by Laughlin [2], who matched the statistics

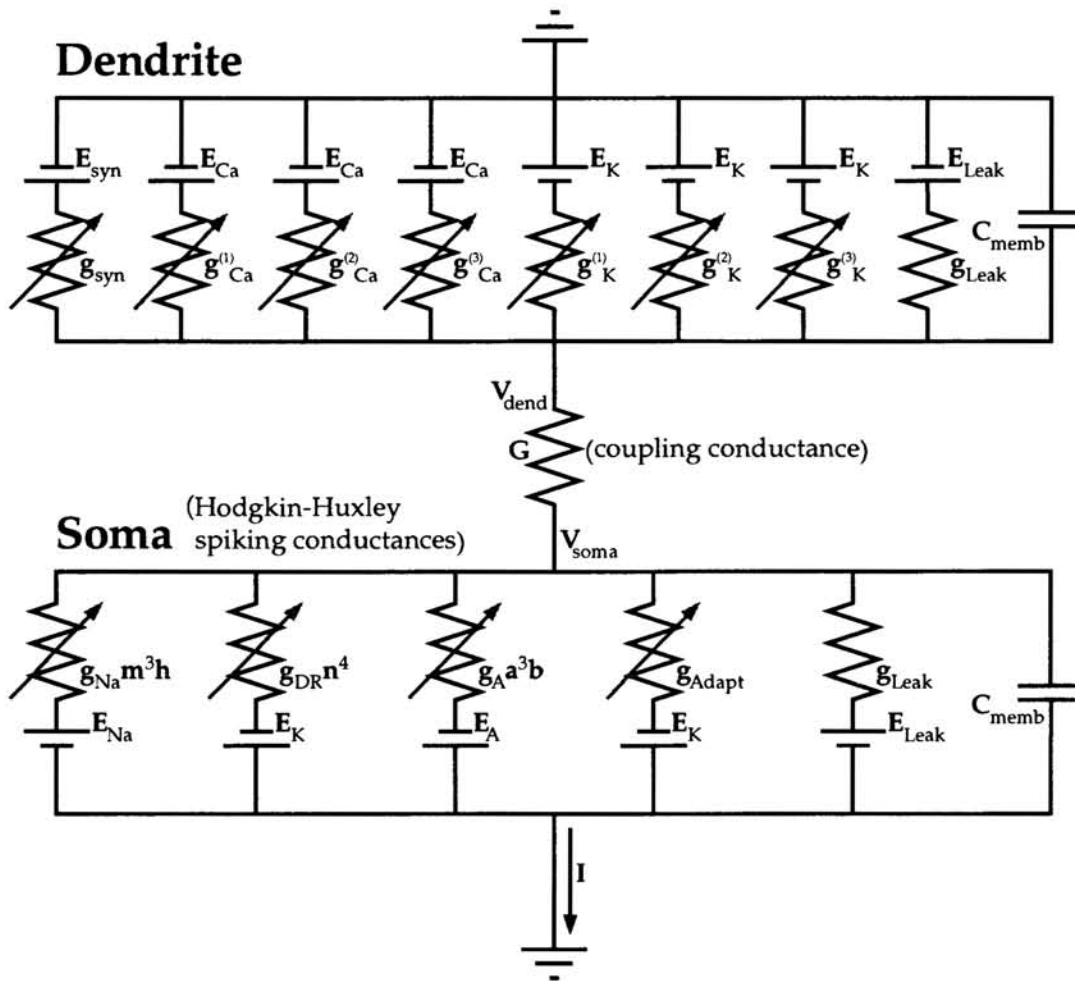

Figure 1: The model neuron contains two compartments to represent the cell's soma and dendrites. To maximize the information transfer, the parameters for six calcium and six potassium voltage-dependent conductances in the dendritic compartment are iteratively adjusted, while the somatic conductances responsible for the cell's spiking behavior are held fixed.

of naturally occurring visual contrasts to the response amplitudes of the blowfly's large monopolar cell.

From a theoretical perspective, the central question is whether a neuron can "learn" the best representation for natural stimuli through experience. During neuronal development, the nature and frequency of incoming stimuli are known to change both the anatomical structure of neurons and the distribution of ionic conductances throughout the cell [3]. We seek a guiding principle that governs the developmental timecourse of the $Na^+$, $Ca^{2+}$ and $K^+$ conductances in the somatic and dendritic membrane by asking how a neuron would set its conductances to transmit as much information as possible. Spiking neurons must associate a range of different inputs to a set of distinct responses—a more difficult task than

keeping the firing rate or excitatory postsynaptic potential (EPSP) amplitude constant under changing conditions, two tasks for which learning rules that change the voltage-dependent conductances have recently been proposed [4, 5]. Learning the proper representation of stimulus information goes beyond simply correlating input and output; an alternative to the classic postulate of Hebb [6], in which synaptic learning in networks is a consequence of correlated activity between pre- and postsynaptic neurons, is required for such learning in a single neuron.

To explore the feasibility of learning rules for information maximization, a simplified model of a neuron consisting of two electrotonic compartments, illustrated in fig. 1, was constructed. The soma (or cell body) contains the classic Hodgkin-Huxley sodium and delayed rectifier potassium conductances, with the addition of a transient potassium "A-"current and an effective calcium-dependent potassium current. The soma is coupled through an effective conductance $G$ to the dendritic compartment, which contains the synaptic input conductance and three adjustable calcium and three adjustable potassium conductances.

The dynamics of this model are given by Hodgkin-Huxley-like equations that govern the membrane potential and a set of activation and inactivation variables, $m_i$ and $h_i$, respectively. In each compartment of the neuron, the voltage $V$ evolves as

$$C\frac{dV}{dt} = \sum_i g_i\, m_i^{p_i}\, h_i^{q_i}\, (E_i - V),\qquad(1)$$

where $C$ is the membrane capacitance, $g_i$ is the (peak) value of the i-th conductance, $p_i$ and $q_i$ are integers, and $E_i$ are the ion-specific reversal potentials. The variables $h_i$ and $m_i$ obey first order kinetics of the type $dm/dt = (m_\infty(V) - m)/\tau(V)$, where $m_\infty(V)$ denotes the steady state activation when the voltage is clamped to $V$ and $\tau(V)$ is a voltage-dependent time constant.

All parameters for the somatic compartment, with the exception of the adaptation conductance, are given by the standard model of Connor $et\ al$ (1977) [7]. This choice of somatic spiking conductances allows spiking to occur at arbitrarily low firing rates. Adaptation is modeled by a calcium-dependent potassium conductance that scales with the firing rate, such that the conductance has a mean value of $34$ mS/cm$^2$ Hz. The calcium and potassium conductances in the dendritic compartment have simple activation and inactivation functions described by distinct Boltzmann functions. Together with the peak conductance values, the midpoint voltages $V_{\frac{1}{2}}$ and slopes $s$ of these Boltzmann functions adapt to the statistics of stimuli. For simplicity, all time constants for the dendritic conductances are set to a constant 5 msec. For additional details and parameter values, see http://www.klab.caltech.edu/infomax.

Hodgkin-Huxley models can exhibit complex behaviors on several timescales, such as firing patterns consisting of "bursts"—sequences of multiple spikes interspersed with periods of silence. We will, however, focus on models of regularly spiking cells that adapt to a sustained stimulus by spiking periodically. To quantify how much information about a continuous stimulus variable $x$ the time-averaged firing rate $f$ of a regularly spiking neuron carries, we use a lower bound [8] on the mutual information $I(f; x)$ between the stimulus

$x$ and the firing rate $f$:

$$I_{\text{LB}}(f; x) = - \int \ln \left( p(f)\, \sigma_f(x) \right) p(x)\, dx - \ln(\sqrt{2\pi e}), \qquad (2)$$

where $p(f)$ is the probability, given the set of all stimuli, of a firing rate $f$, and $\sigma_f^2(x)$ is the variance of the firing rate in response to a given stimulus $x$.

To maximize the information transfer, does a neuron need to "know" the arrival rates of photons impinging on the retina or the frequencies of sound waves hitting the ear's tympanic membrane? Since the ion channels in the dendrites only sense a voltage and not the stimulus directly, the answer to this question, fortunately, is no: maximizing the information between the firing rate $f$ and the dendritic voltage $V_{\text{dend}}(t)$ is equivalent to maximizing the information about the stimuli, as long as we can guarantee that the transformation from stimuli to firing rates is always one-to-one.

Since a neuron must be able to adapt to a changing environment and shifting intra- and extracellular conditions [4], learning and relearning of the proper conductance parameters, such as the channel densities, should occur on a continual basis. An alphabet zoo of different calcium ($Ca^{2+}$) conductances in neurons of the central nervous system, denoted 'L', 'N', 'P', 'R', and 'T'-conductances, reflects a wealth of different voltage and pharmacological properties [9], matching an equal diversity of potassium ($K^+$) channels. No fewer than ten different genes code for various $Ca^{2+}$ subunits, allowing for a combinatorial number of functionally different channels [10]. A self-regulating neuron should be able to express different ionic channels and insert them into the membrane. In information maximization, the parameters for each of the conductances, such as the number of channels, are continually modified in the direction that most increases the mutual information $I[f; V_{\text{dend}}(t)]$ each time a stimulus occurs.

The standard approach to such a problem is known as stochastic approximation of the mutual information, which was recently applied to feedforward neural networks for blind source sound separation by Bell and Sejnowski [11]. We define a "free energy" $\mathcal{F} = E(f) - \beta^{-1} I_{\text{LB}}(f; x)$, where $E(f)$ incorporates constraints on the peak or mean firing rate $f$, and $\beta$ is a Lagrangean parameter that balances the mutual information and constraint satisfaction. Stochastic approximation then consists of adjusting the parameter r of a voltage-dependent conductance by

$$\Delta r|_x = -\eta \frac{\partial}{\partial r} \frac{\delta \mathcal{F}(x)}{\delta p(x)} \qquad (3)$$

whenever a stimulus $x$ is presented; this will, by definition, occur with probability $p(x)$. In the model, the stimuli are taken to be maintained synaptic input conductances $g_{\text{syn}}$ lasting 200 msec and drawn randomly from a fixed, continuous probability distribution. After an initial transient, we assume that the voltage waveform $V_{\text{dend}}(t)$ settles into a simple periodic limit cycle as dictated by the somatic spiking conductances. We thus posit the existence of the invertible composition of maps, such that the input conductance $g_{\text{syn}}$ maps onto a periodic voltage waveform $V_{\text{dend}}(t)$ of period $T$, from thence onto an averaged current $\langle I \rangle = 1/T \int_0^T I(t)\, dt$ to the soma, and then finally onto an output firing rate $f$. The last element in this chain of transformations, the steady-state current-discharge

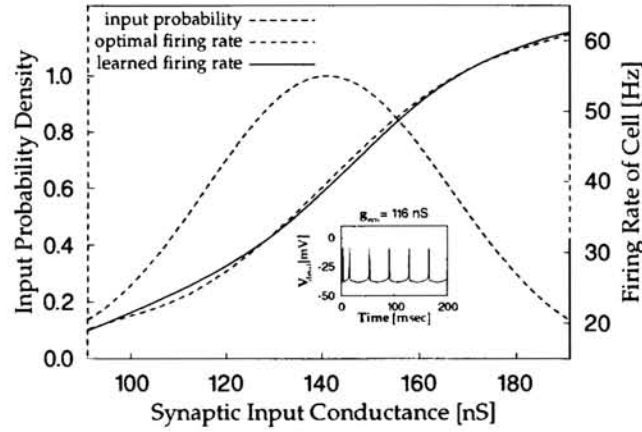

Figure 2: The inputs to the model are synaptic conductances, drawn randomly from a Gaussian distribution of mean 141 nS and standard deviation of 25 nS with the restriction that the conductance be non-negative (dot-dashed line). The learning rule in eq. 4—maximizing the information in the cell's firing rate—was used to adjust the peak conductances, midpoint voltages, and slopes of the "dendritic" $Ca^{2+}$ and $K^+$ conductances over the course of 10.9 (simulated) minutes. . The learning rate decayed with time: $\eta(t) = \eta_0 \exp(-t/\tau_{\text{learning}})$, with $\eta_0 = 4.3 \times 10^{-3}$ and $\tau_{\text{learning}} = 4.4$ sec. The optimal firing rate response curve (dotted line) is asymptotically proportional to the cumulative probability distribution of inputs. The inset illustrates the typical timecourse of the dendritic voltage in the trained model.

relationship at the soma, can be predicted from the theory of dynamical systems (see http://www.klab.caltech.edu/~stemmler for details).

The voltage and the conductances are nonlinearly coupled: the conductances affect the voltage, which, in turn, sets the conductances. Since the mutual information is a *global* property of the stimulus set, the learning rule for any one conductance would depend on the values of all other conductances, were it not for the nonlinear feedback loop between voltages and conductances. This nonlinear coupling must satisfy the strict physical constraint of charge conservation: when the neuron is firing periodically, the average current injected by the synaptic and voltage-dependent conductances must equal the average current discharged by the neuron. Remarkably, charge conservation results in a learning mechanism that is strictly *local*, so that the mechanism for changing one conductance does not depend on the values of any other conductances.

For instance, information maximization predicts that the peak calcium or potassium conductance $g_i$ changes by

$$\Delta g_i = \eta(t) \left\langle \frac{\delta}{\delta V_{\text{dend}}(t)} \langle m_i h_i (E_i - V_{\text{dend}}) \rangle + c(\langle V_{\text{dend}} \rangle) \, m_i h_i (E_i - V_{\text{dend}}) \right\rangle \qquad (4)$$

each time a stimulus is presented. Here $\eta(t)$ is a time-dependent learning rate, the angular brackets indicate an average over the stimulus duration, and $c(\langle V_{\text{dend}} \rangle)$ is a simple function that is zero for most commonly encountered voltages, equal to a positive constant below some minimum, and equal to a negative constant above some maximum voltage. This

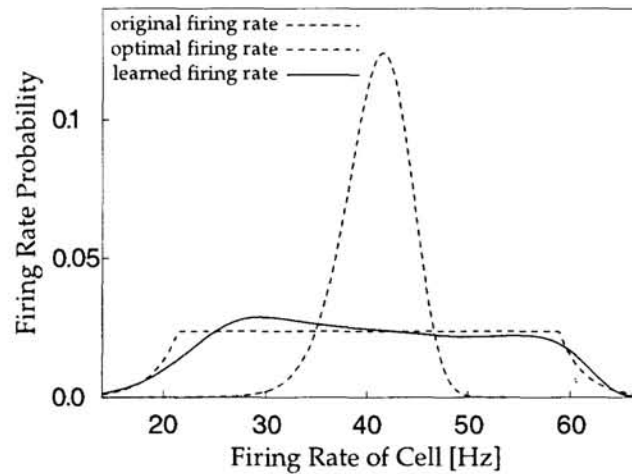

Figure 3: The probability distribution of firing rates before and after adaptation of voltage-dependent conductances. Learning shifts the distribution from a peaked distribution to a much flatter one, so that the neuron uses each firing rate within the range [22, 59] Hz equally often in response to randomly selected synaptic inputs.

function represents the constraint on the maximum and minimum firing rate, which sets the limit on the neuron's dynamic range. A constraint on the mean firing rate implies that $c(\langle V_{\text{dend}} \rangle)$ is simply a negative constant for all suprathreshold voltages. Under this constraint, the optimal distribution of firing rates becomes exponential (not shown). This latter case corresponds to transmitting as much information as possible in the rate while firing as little as possible.

Given a stimulus $x$, the dominant term $\delta/\delta V(t) \langle m_i h_i (E_i - V) \rangle$ of eq. 4 changes those conductances that increase the slope of the firing rate response to $x$. A higher slope means that more of the neuron's limited range of firing rates is devoted to representing the stimulus $x$ and its immediate neighborhood. Since the learning rule is democratic yet competitive, only the most frequent inputs "win" and thereby gain the largest representation in the output firing rate.

In Fig. 2, the learning rule of eq. 4—generalized to also change the midpoint voltage and steepness of the activation and inactivation functions—has been used to train the model neuron as it responds to random, 200 msec long amplitude modulations of a synaptic input conductance to the dendritic compartment. The cell "learns" the statistical structure of the input, matching its adapted firing rate to the cumulative distribution function of the conductance inputs. The distribution of firing rates shifts from a peaked distribution to a much flatter one, so that all firing rates are used nearly equally often (Fig. 3). The information in the firing rate increases by a factor of three to 10.7 bits/sec, as estimated by adding a 5 msec, Gaussian-distributed noise jitter to the spike times.

Changing how tightly the stimulus amplitudes are clustered around the mean will increase or decrease the slope of the firing rate response to input, without necessarily changing the average firing rate. Neuronal systems are known to adapt not only to the mean of

the stimulus intensity, but also to the variance of the stimulus [12]. We predict that such adaptation to stimulus variance will occur not just at the level of networks of neurons, but also at the single cell level.

While the detailed substrate for maximizing the information at both the single cell and network level awaits experimental elucidation, the terms in the learning rule of eq. 4 have simple biophysical correlates: the derivative term, for instance, is reflected in the stochastic flicker of ion channels switching between open and closed states. The transitions between simple open and closed states will occur at a rate proportional to $(\delta/\delta V \langle m(V) \rangle)^{\gamma}$ in equilibrium, where the exponent $\gamma$ is $1/2$ or $1$, depending on the kinetic model. To change the information transfer properties of the cell, a neuron could use state-dependent phosphorylation of ion channels or gene expression of particular ion channel subunits, possibly mediated by G-protein initiated second messenger cascades, to modify the properties of voltage-dependent conductances. The tools required to adaptively compress information from the senses are thus available at the subcellular level.

# References

[1] D. L. Ruderman, *Network* **5**(4), 517 (1995), R. J. Baddeley and P. J. B. Hancock, *Proc. Roy. Soc. B* **246**, 219 (1991), J. J. Atick, *Network* **3**, 213 (1992).

[2] S. Laughlin, *Z. Naturforsch.* **36**c, 910 (1981).

[3] Purves, D. *Neural activity and the growth of the brain*, (Cambridge University Press, NY, 1994); X. Gu and N. C. Spitzer, *Nature* **375**, 784 (1995).

[4] G. LeMasson, E. Marder, and L. F. Abbott, *Science* **259**, 1915 (1993).

[5] A. J. Bell, *Neural Information Processing Systems* **4**, 59 (1992).

[6] D. O. Hebb, *The Organization of Behavior* (Wiley, New York, 1949).

[7] J. A. Connor, D. Walter, R. McKown, *Biophys. J.* **18**, 81 (1977).

[8] R. B. Stein, *Biophys. J.* **7**, 797 (1967).

[9] R. B. Avery and D. Johnston, *J. Neurosci.* **16**, 5567 (1996), F. Helmchen, K. Imoto, and B. Sakmann, *Biophys. J.* **70**, 1069 (1996).

[10] F. Hofmann, M. Biel, and V. Flockerzi, *Ann. Rev. Neurosci.* **17**, 399 (1994).

[11] Y. Z. Tsypkin, *Adaptation and Learning in Automatic Systems* (Academic Press, NY, 1971)], R. Linsker, *Neural Comp.* **4**, 691 (1992), and A. J. Bell and T. J. Sejnowski, *Neural Comp.* **7**, 1129 (1995).

[12] S. M. Smirnakis *et al.*, *Nature* **386**, 69 (1997).
